# Portmanteau Vocabularies for Multi-Cue Image Representation

**Fahad Shahbaz Khan**[1]**, Joost van de Weijer**[1]**, Andrew D. Bagdanov**[1,2]**, Maria Vanrell**[1]

[1]Centre de Visio per Computador, Computer Science Department
[1]Universitat Autonoma de Barcelona, Edifci O, Campus UAB (Bellaterra), Barcelona, Spain
[2] Media Integration and Communication Center, University of Florence, Italy

## Abstract

We describe a novel technique for feature combination in the bag-of-words model of image classification. Our approach builds discriminative compound words from primitive cues learned independently from training images. Our main observation is that modeling joint-cue distributions independently is more statistically robust for typical classification problems than attempting to empirically estimate the dependent, joint-cue distribution directly. We use Information theoretic vocabulary compression to find discriminative combinations of cues and the resulting vocabulary of *portmanteau*[1] words is compact, has the cue binding property, and supports individual weighting of cues in the final image representation. State-of-the-art results on both the Oxford Flower-102 and Caltech-UCSD Bird-200 datasets demonstrate the effectiveness of our technique compared to other, significantly more complex approaches to multi-cue image representation.

## 1 Introduction

Image categorization is the task of classifying an image as containing an objects from a predefined list of categories. One of the most successful approaches to this problem is the bag-of-words (BOW) [4, 15, 11, 2]. In the bag-of-words model an image is first represented by a collection of local image features detected either sparsely or in a regular, dense grid. Each local feature is then represented by one or more cues, each describing one aspect of a small region around the corresponding feature. Typical local cues include color, shape, and texture. These cues are then quantized into visual words and the final image representation is a histogram over these visual vocabularies. In the final stage of the BOW approach the histogram representations are sent to a classifier.

The success of BOW is highly dependent on the quality of the visual vocabulary. In this paper we investigate visual vocabularies which are used to represent images whose local features are described by both shape and color. To extend BOW to multiple cues, two properties are especially important: cue binding and cue weighting. A visual vocabulary is said to have the *binding property* when two independent cues appearing at the same location in an image remain coupled in the final image representation. For example, if every local patch in an image is independently described by a shape word and a color word, in the final image representation using compound words the binding property ensures that shape and color words coming from the same feature location are coupled in the final representation. The term *binding* is borrowed from the neuroscience field where it is used to describe the way in which humans select and integrate the separate cues of objects in the correct combinations in order to accurately recognize them [17]. The property of *cue weighting* implies that it is possible

to adapt the relevance of each cue depending on the dataset. The importance of cue weighting can be seen from the success of Multiple Kernel Learning (MKL) techniques where weights for each cue are automatically learned [3, 13, 21, 14, 1, 20].

Traditionally, two approaches exist to handle multiple *cues* in BOW. When each cue has its own visual vocabulary the result is known as a *late fusion* image representation in which an image is represented as one histogram over shape-words and another histogram over color-words. Such a representation does not have the cue binding property, meaning that it is impossible to know exactly which color-shape events co-occurred at local features. Late fusion does, however, allow cue weighting. Another approach, called *early fusion*, constructs a single visual vocabulary of joint color-shape words. Representations over early fusion vocabularies have the cue binding property, meaning that the spatial co-occurrence of shape and color events is preserved. However, cue weighting in early fusion vocabularies is very cumbersome since must be performed before vocabulary construction making cross-validation very expensive. Recently, Khan et al. [10] proposed a method which combines cue binding and weighting. However, their final image representation size is equal to number of vocabulary words times the number of classes, and is therefore not feasible for the large data sets considered in this paper.

A straightforward, if combinatorially inconvenient, approach to ensuring the binding property is to create a new vocabulary that contains one word for each combination of original shape and color feature. Considering that each of the original shape and color vocabularies may contain thousands of words, the resulting joint vocabulary may contain millions. Such large vocabularies are impractical as estimating joint color-shape statistics is often infeasible due to the difficulty of sampling from limited training data. Furthermore, with so many parameters the resulting classifiers are prone to overfitting. Because of this and other problems, this type of joint feature representation has not been further pursued as a way of ensuring that image representations have the binding property.

In recent years a number of vocabulary compression techniques have appeared that derive small, discriminative vocabularies from very large ones [16, 7, 5]. Most of these techniques are based on information theoretic clustering algorithms that attempt to combine words that are equivalently discriminative for the set of object categories being considered. Because these techniques are guided by the discriminative power of clusters of visual words, estimates of class-conditional visual word probabilities are essential. These recent developments in vocabulary compression allow us to reconsider the direct, Cartesian product approach to building compound vocabularies.

These vocabulary compression techniques have been demonstrated on single-cue vocabularies with a few tens of thousands of words. Starting from even moderately sized shape and color vocabularies results in a compound shape-color vocabulary an order of magnitude larger. In such cases, robust estimates of the underlying class-conditional joint-cue distributions may be difficult to obtain. We show that for typical datasets a strong independence assumption about the joint color-shape distribution leads to more robust estimates of the class-conditional distributions needed for vocabulary compression. In addition, our estimation technique allows flexible cue-specific weighting that cannot be easily performed with other cue combination techniques that maintain the binding property.

## 2   Portmanteau vocabularies

In this section we propose a new multi-cue vocabulary construction method that results in compact vocabularies which possess both the cue binding and the cue weighting properties described above. Our approach is to build *portmanteau vocabularies* of discriminative, compound shape and color words chosen from independently learned color and shape lexicons. The term portmanteau is used in natural language for words which are a blend of two other words and which combine their meaning. We use the term *portmanteau* to describe these compound terms to emphasize the fact that, similarly to the use of neologistic portmanteaux in natural language to capture complex and compound concepts, we create groups of color and shape words to describe semantic concepts inadequately described by shape or color alone.

A simple way to ensure the binding property is by considering a product vocabulary that contains a new word for every combination of shape and color terms. Assume that $S = \{s_1, s_2, ..., s_M\}$ and $C = \{c_1, c_2, ..., c_N\}$ represent the visual shape and color vocabularies, respectively. Then the

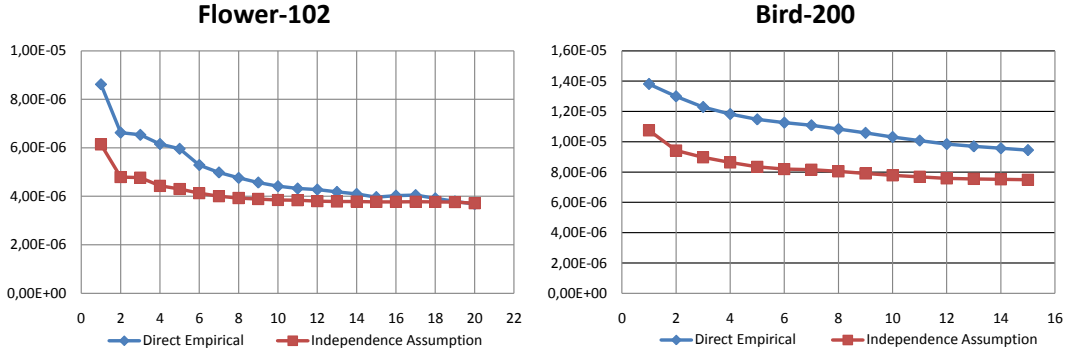

Figure 1: Comparison of two estimates of the joint cue distribution $p(S, C|R)$ on two large datasets. The graphs plot the Jenson-Shannon divergence between each estimate and the true joint distribution as a functions of the number of training images used to estimate them. The true joint distribution is estimated empirically over all images in each dataset. Estimation using the independence assumption of equation (2) yields similar or better estimates than their empirical counterparts.

product vocabulary is given by

$$W = \{w_1, w_2, ..., w_T\} = \{\{s_i, c_j\} \mid 1 \le i \le M, 1 \le j \le N\},$$

where $T = M \times N$. We will also use the the notation $s_m$ to identify a member from the set $S$.

A disadvantage of vocabularies of compound terms constructed by considering the Cartesian product of all primitive shape and color words is that the total number of visual words is equal to the number of color words times the number of shape words, which typically results in hundreds of thousands of elements in the final vocabulary. This is impractical for two reasons. First, the high dimensionality of the representation hampers the use of complex classifiers such as SVMs. Second, insufficient training data often renders robust estimation of parameters very difficult and the resulting classifiers tend to overfit the training set. Because of these drawbacks, compound product vocabularies have, to the best of our knowledge, not been pursued in literature. In the next two subsections we discuss our approach to overcoming these two drawbacks.

## 2.1 Compact Portmanteau Vocabularies

In recent years, several algorithms for feature clustering have been proposed which compress large vocabularies into small ones [16, 7, 5]. To reduce the high-dimensionality of the product vocabulary, we apply Divisive Information-Theoretic feature Clustering (DITC) algorithm [5], which was shown to outperform AIB [16]. Furthermore, DITC has also been successfully employed to construct compact pyramid representations [6].

The DITC algorithm is designed to find a fixed number of clusters which minimize the loss in mutual information between clusters and the class labels of training samples. In our algorithm, loss in mutual information is measured between original product vocabulary and the resulting clusters. The algorithm joins words which have similar discriminative power over the set of classes in the image categorization problem. This is measured by the probability distributions $p(R|w_t)$, where $R = \{r_1, r_2, ..r_L\}$ is the set of $L$ classes.

More precisely, the drop in mutual information $I$ between the vocabulary $W$ and the class labels $R$ when going from the original set of vocabulary words $W$ to the clustered representation $W^R = \{W_1, W_2, ..., W_J\}$ (where every $W_j$ represents a cluster of words from $W$) is equal to

$$I(R; W) - I(R; W^R) = \sum_{j=1}^{J} \sum_{w_t \in W_j} p(w_t) KL(p(R|w_t) \,||\, p(R|W_j)), \qquad (1)$$

where KL is the Kullback-Leibler divergence between two distributions. Equation (1) states that the drop in mutual information is equal to the prior-weighted KL-divergence between a word and its assigned cluster. The DITC algorithm minimizes this objective function by alternating computation

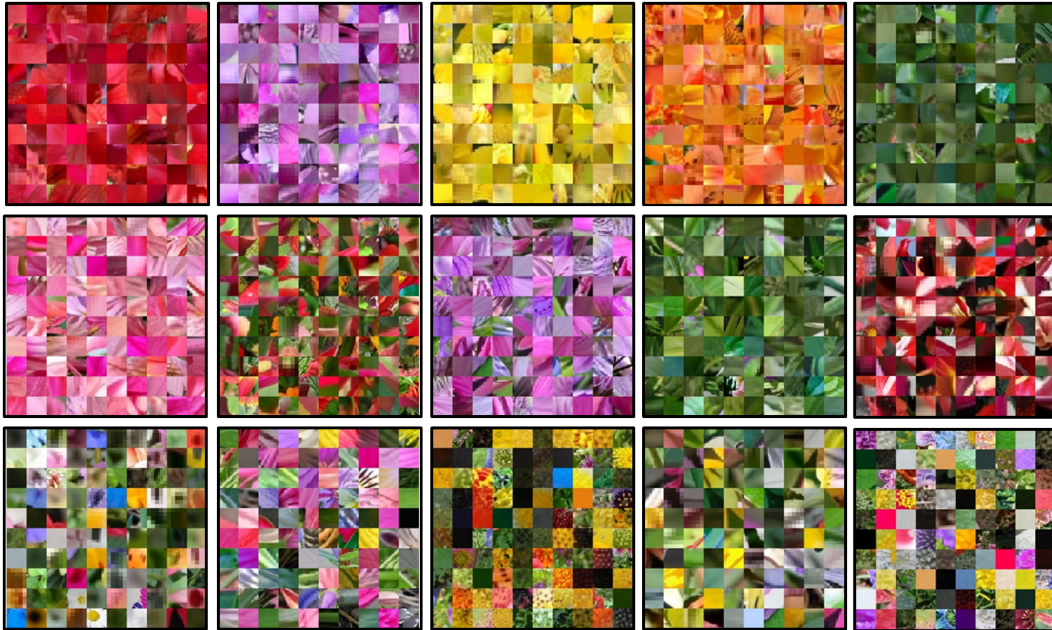

Figure 2: The effect of $\alpha$ on DITC clusters. Each of the large boxes contains 100 image patches sampled from one Portmanteau word on the Oxford Flower-102 dataset. Top row: five clusters for $\alpha = 0.1$. Note how these clusters are relatively homogeneous in color, while shape varies considerably within each. Middle row: five clusters sampled for $\alpha = 0.5$. The clusters show consistency over both color and shape. Bottom row: five clusters sampled for $\alpha = 0.9$. Notice how in this case shape is instead homogeneous within each cluster.

of the cluster distributions and assignment of compound visual words to their closest cluster. For more details on the DITC algorithm we refer to Dhillon et al. [5]. Here we apply the DITC algorithm to reduce the high-dimensionality of the compound vocabularies. We call the compact vocabulary which is the output of the DITC algorithm the *portmanteau vocabulary* and its words accordingly *portmanteau words*. The final image representation $p(W^R)$ is a distribution over the portmanteau words.

## 2.2 Joint distribution estimation

In solving the problem of high-dimensionality of the compound vocabularies we seemingly further complicated the estimation problem. As DITC is based on estimates of the class-conditional distributions $p(S, C|R) = p(W|R)$ over product vocabularies, we have increased the number of parameters to be estimated to $M \times N \times L$. This can easily reach millions of parameters for standard image datasets. To solve this problem we propose to estimate the class conditional distributions by assuming independence of color and shape, given the class:

$$p(s_m, c_n|R) \propto p(s_m|R)p(c_n|R). \qquad (2)$$

Note that we do not assume independence of the cues themselves, but rather the less restrictive independence of the cues given the class. Instead of directly estimating the empirical joint distribution $p(S, C|R)$, we reduce the number of parameters to estimate to $(M + N) \times L$, which in the vocabulary configurations discussed in this paper represents a reduction in complexity of two orders of magnitude. As an additional advantage, we will show in section 2.3 that estimating the joint distribution $p(S, C|R)$ allows us to introduce cue weighting.

To verify the quality of the empirical estimates of equation (2) we perform the following experiment. In figure 1 we plot the Jensen-Shannon (JS) divergence between the empirical joint distribution obtained from the test images and the two estimates: direct estimation of the empirical joint distribution $p(S, C|R)$ on the training set, and an approximate estimate made by assuming independence as in

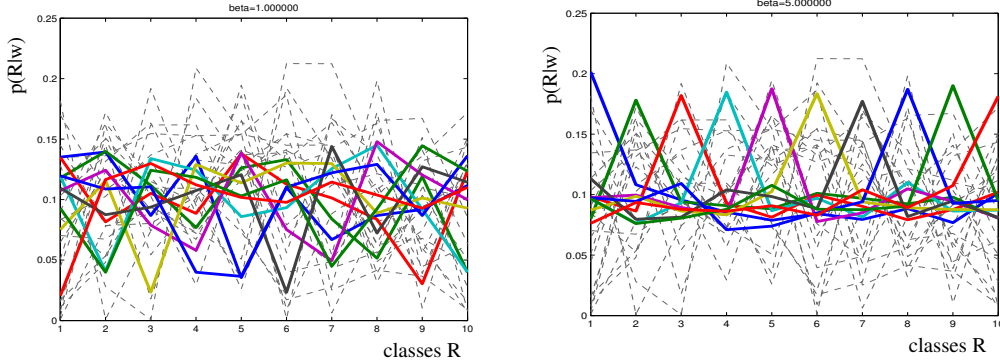

Figure 3: The effect of $\beta$ on DITC clusters. For 20 words $p\left(R|w_t\right)$ is plotted in dotted grey lines. DITC is used to obtain ten portmanteau means $p\left(R|W_j\right)$ are plotted in different colors. On the left is shown the final clustering for $\beta = 1.0$. Note that none of the portmanteau means are especially discriminative for one particular class. On the right, however, for $\beta = 5.0$ each portmanteau concentrates on discriminating one class.

equation (2). Results are provided as a function of the number of training images for two large datasets. A low JS-divergence means a better estimate of the true joint-cue distribution. The plotted lines show the curves for a color cue vocabulary of 100 words and a shape cue vocabulary of 5,000 words, resulting in a product vocabulary of 500,000 words. On both datasets we see that the independence assumption actually leads to a better or equally good estimate of the joint distribution. Increasing the number of training samples, or starting with smaller color and shape vocabularies and hence reducing the number of parameters to estimate, will improve direct empirical estimates of $p(S, C)$. However, figure 1 shows that for typical vocabulary settings on large datasets the independence assumption results in equivalently good or better estimates of the joint distribution.

## 2.3   Cue weighting

Constructing the compact portmanteau vocabularies based on the independence assumption significantly reduces the number of parameters to estimate. Furthermore, as we will see in this section, it allows us to control the relative contribution of color and shape cues in the final representation.

We introduce a weighting parameter $\alpha \in [0, 1]$ in the estimate of $p(C, S)$:

$$p^{\alpha}(s_m, c_n|R) \propto p(s_m|R)^{\alpha} p(c_n|R)^{1-\alpha} \tag{3}$$

where an $\alpha$ close to zero results in a larger influence of the color words, and a $\alpha$ close to one leads to a vocabulary which focuses predominantly on shape.

To illustrate the influence of $\alpha$ on the vocabulary construction, we show samples from portmanteau words obtained on the Oxford Flower-102 dataset (see figure 4) in figure 2. The DITC algorithm is applied to reduce the product vocabulary of 500,000 compound words to 100 portmanteau words. For settings of $\alpha \in \{0.1, 0.5, 0.9\}$ we show five of the hundred words. Each word is represented by one hundred randomly sampled patches from the dataset which have been assigned to the word. The effect of changing the $\alpha$ can be clearly seen. For low $\alpha$ the Portmanteau words exhibit homogeneity of color but lack within-cluster shape consistency. On the other hand for high $\alpha$ the words show strong shape homogeneity such as low and high frequency lines and blobs, while color is more uniformly distributed. For a setting of $\alpha = 0.5$ the clustering is more consistent in both color and shape.

Additionally, another parameter $\beta$ is introduced:

$$p^{\alpha,\beta}(s_m, c_n|R) \propto \left(p(s_m|R)^{\alpha} p(c_n|R)^{1-\alpha}\right)^{\beta} \tag{4}$$

To illustrate the influence of $\beta$ consider the following experiment on synthetic data. We generate a set of 100 words which have random discriminative power $p\left(R|w_t\right)$ over $L = 10$ classes. In figure 3

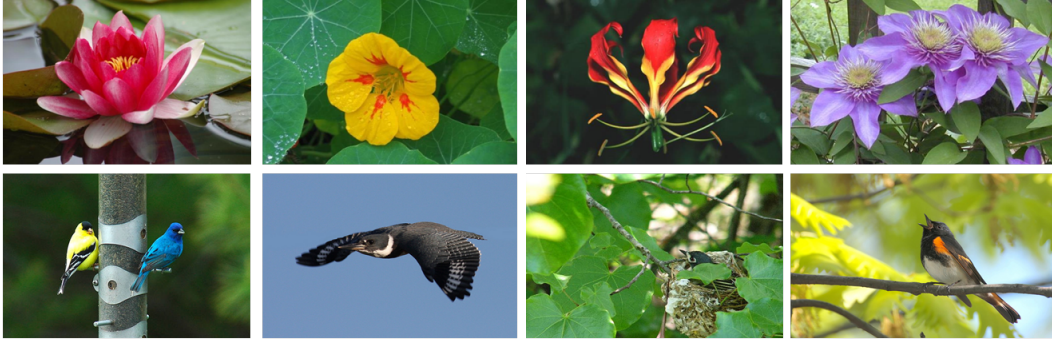

Figure 4: Example images from the two datasets used in our experiments Top: images from four categories of the Flower-102 dataset. Bottom: four example images from the Bird-200 dataset.

we show the $p\left(R|w_t\right)$ for a subset of 20 words in grey, and $p\left(R|W_j\right) \propto \sum\limits_{w_t \in W_j} p(w_t)p(R|w_t)$ for the ten portmanteau words in color. We observe that increasing the $\beta$ parameter directs DITC to find clusters which are each highly discriminative for a single class, rather than being discriminative over all classes. We found that higher $\beta$ values often lead to image representations which improve classification results.

These weighting parameters are learned through cross validation on the training set. In practice we found $\alpha$ to change with the data set according to the importance of color and shape. The $\beta$ parameter was found to to be constant at a value 5 for the two datasets evaluated in this paper. Both parameters were found to significantly improve results on both datasets.

## 2.4 Image representation with portmanteau vocabularies

We summarize our approach to constructing portmanteau vocabularies for image representation. We emphasize the fact that our approach is fundamentally about deriving compact multi-cue image representations and, as such, can be used as a drop-in replacement in any bag-of-words pipeline.

Image representation by portmanteau vocabulary built from color and shape cues follows these steps:

1. Independent color and shape vocabularies are constructed by standard K-means clustering over color and shape descriptors extracted from training images.

2. Empirical class-conditional word distributions $p(S|R)$ and $p(C|R)$ are computed from the training set, the joint cue distribution $P(S,C|R)$ is estimated assuming conditional independence as in equation (4).

3. The portmanteau vocabulary is computed with the DITC algorithm. The output of the DITC is a list of indexes which, for each member of the compound vocabulary maps to one of the $J$ portmanteau words.

4. Using the index list output by DITC, the original image features are revisited and the index corresponding the compound shape-color word at each feature is used to represent each image as a histogram over the portmanteau vocabulary.

## 3 Experimental results

We follow the standard bag-of-words approach. We use a combination of interest-point detectors along with a dense multi-scale grid detector. The SIFT descriptor [12] is used to construct a shape vocabulary. For color we use the color name descriptor, which is computed by converting sRGB values to color names according to [19] after which each patch is represented as a histogram over the eleven color names. The shape and color vocabularies are constructed using the standard K-means algorithm. In all our experiments we use a shape vocabulary of 5000 words and a color vocabulary of 100 words. Applying Laplace weighting was not found to influence the results and

therefore not used in the experiments. The classifier is a non-linear, multi-way, one-versus-all SVM using the $\chi^2$ kernel [24]. Each test image is assigned the label of the classifier giving the highest response and the final classification score is the mean recognition rate per category.

We performed several experiments to validate our approach to building multi-cue vocabularies by comparing with other methods which are based on exactly the same initial SIFT and CN descriptors:

- **Shape and Color only**: a single vocabulary of 5000 SIFT words and one of 100 CN words.
- **Early fusion**: SIFT and CN are concatenated into single descriptor. The relative weight of shape and color is optimized by cross-validation. Note that cross-validation on cue weighting parameters for early fusion must be done over the *entire* BOW pipeline, from vocabulary construction to classification. Vocabulary size is 5000.
- **Direct empirical**: DITC based on the empirical distribution of $p(S, C|R)$ over a total of 500.000 compound words estimated on the training set.
- **Independence assumption**: where $p(S, C|R) = p(S|R)p(C|R)$ is assumed. We also show separate results with and without using $\alpha$ and $\beta$.

In all cases the color-shape visual vocabularies are compressed to 500 visual words and spatial pyramids are constructed for the final image representation as in [11]. All of the above approaches were evaluated on two standard and challenging datasets: Oxford Flower-102 and Caltech-UCSD Bird-200. The train-test splits are fixed for both datasets and are provided on their respective websites.[2]

## 3.1  Results on the Flower-102 and Bird-200 datasets

The Oxford Flower-102 dataset contains 8189 images of 102 different flower species. It is a challenging dataset due to significant scale and illumination changes (see figure 4). The results are presented in table 1(a). We see that shape alone yields results superior to color. Early fusion is reasonably good at $70.5\%$. This is however obtained through laborious cross validation to obtain the optimal balance between CN and SIFT cues. Since our cue weighting is done after the initial vocabulary and histogram construction, cross-validation is significantly faster than for early fusion.

The bottom three rows of table 1(a) give the results of our approach to image representation with portmanteau vocabularies in a variety of configurations. The direct empirical estimation of the joint shape-color distribution provides slightly better results than estimation based on the independence assumption. However, weighting the two visual cues using the $\alpha$ parameter described in equation (3) in the independent estimation of $p(s, c|\text{class})$ improves the results significantly. In particular, the gain of almost $7\%$ obtained by adding $\beta$ is remarkable. The best recognition performance were obtained for $\alpha = 0.8$ and $\beta = 5$.

The Caltech-UCSD Bird-200 dataset contains 6033 images from 200 different bird species. This dataset contains many bird species that closely resemble each other in terms of color and shape cues, making the recognition task extremely difficult. Table 1(a) contains test results for our approach on Bird-200 as well. Interestingly, on this dataset color outperforms shape alone and early fusion yields only a small improvement over color. Results based on portmanteau vocabularies outperform early fusion, and estimation based on the independence assumption provide better results than direct empirical estimation. These results are further improved by the introduction of cue weighting with a final score of $22.4\%$ obtained with $\alpha = 0.7$ and $\beta = 5$ outperforming all others.

## 3.2  Comparison with the state-of-the-art

Recently, an extensive performance evaluation of color descriptors was presented by van de Sande et al. [18]. In this evaluation the OpponentSIFT and C-SIFT were reported to provide superior performance on image categorization problems. We construct a visual vocabulary of 5000 visual words for both OpponentSIFT and C-SIFT and apply the DITC algorithm to compress it to 500 visual words. As shown in table 1(b), Our approach provides significantly better results compared to both OpponentSIFT and C-SIFT, possibly due to the fact neither supports cue weighting.

| Method | Flower-102 | Bird-200 |
|---|---|---|
| Shape only | 60.7 | 12.9 |
| Color only | 48.5 | 16.8 |
| Early Fusion | 70.5 | 17.0 |
| Direct empirical | 64.6 | 18.9 |
| Independent | 63.5 | 19.8 |
| Independent + $\alpha$ | 66.4 | 21.6 |
| Independent + $\alpha$ + $\beta$ | **73.3** | **22.4** |

(a)

| Method | Bird-200 | Flower-102 |
|---|---|---|
| OpponentSIFT | 14.0 | 69.2 |
| C-SIFT | 13.9 | 65.9 |
| MKL [13] | – | 72.8 |
| MKL [3] | 19.0 | – |
| Random Forest [23] | 19.2 | – |
| Saliency [9] | – | 71.0 |
| Our Approach | **22.4** | **73.3** |

(b)

Table 1: Comparative evaluation of our approach. (a) Classification score on Flower-102 and Bird-200 datasets for individual features, early fusion and several configurations of our approach. (b) Comparison of our approach to the state-of-the-art on the Bird-200 and Flower-102 datasets.

In recent years, combining multiple cues using Multiple Kernel Learning (MKL) techniques has received a lot of attention. These approaches combine multiple cues and multiple kernels and apply per-class cue weighting. Table 1(b) includes two recent MKL techniques that report state-of-the-art performance. The technique described in [3] is based on geometric blur, grayscale SIFT, color SIFT and full image color histograms, while the approach in [13] also employs HSV, SIFT int, SIFT bd, and HOG descriptors in the MKL framework of [21]. Despite the simplicity of our approach, which is based on only two cues and a single kernel, it outperforms these complex multi-cue learning techniques. Also note that both MKL approaches are based on learning class-specific weighting for multiple cues. This is especially cumbersome when there exist several hundred object categories in a dataset (e.g. the Bird-200 dataset contains 200 bird categories). In contrast to these approaches, we learn a global, class-independent cue weighting parameters to balance color and shape cues.

On the Flower-102 dataset, our final classification score of 73.3% is comparable to the state-of-the-art recognition performance [13, 9, 8][3] obtained on this dataset. It should be noted that Nilsback and Zisserman [13] obtain a classification performance of 72.8% using segmented images and a combination of four different visual cues in a multiple kernel learning framework. Our performance, however, is obtained on unsegmented images using only color and shape cues. On the Bird-200 dataset, our approach significantly outperforms state-of-the-art methods [23, 3, 22].

## 4   Conclusions

In this paper we propose a new method to construct multi-cue, visual *portmanteau* vocabularies that combine color and shape cues. When constructing a multi-cue vocabulary two properties are especially desirable: cue binding and cue weighting. Starting from multi-cue product vocabularies we compress this representation to form discriminative compound terms, or portmanteaux, used in the final image representation. Experiments demonstrate that assuming independence of visual cues given the categories provides a robust estimation of joint-cue distributions compared to direct empirical estimation. Assuming independence also has the advantage of both reducing the complexity of the representation by two orders of magnitude and allowing flexible cue weighting. Our final image representation is compact, maintains the cue binding property, admits cue weighting and yields state-of-the-art performance on the image categorization problem.

We tested our approach on two datasets, each with more than one hundred object categories. Results demonstrate the superiority of our approach over existing ones combining color and shape cues. We obtain a gain of 2.8% and 5.4% over the early fusion approach. Our approach also outperforms methods based on multiple cues and MKL with per-class parameter learning. This leaves open the possibility of using our approach to multi-cue image representation within an MKL framework.

**Acknowledgments**: This work is supported by the EU project ERG-TS-VICI-224737; by the Spanish Research Program Consolider-Ingenio 2010: MIPRCV (CSD200700018); by the Tuscan Regional project MNEMOSYNE (POR-FSE 2007-2013, A.IV-OB.2); and by the Spanish projects TIN2009-14173, TIN2010-21771-C02-1. Joost van de Weijer acknowledges the support of a Ramon y Cajal fellowship.

## Footnotes

[1]A *portmanteau* is a combination of two or more words to form a neologism that communicates a concept better than any individual word (e.g. Ski resort + Konference = *Skonference*). We use the term to describe our vocabularies to emphasize the connotation with combining color and shape words into new, more meaningful representations.

[2]The Flower-102 dataset at `http://www.robots.ox.ac.uk/vgg/research/flowers/` and the Birds-200 set at `http://www.vision.caltech.edu/visipedia/CUB-200.html`

[3]From correspondence with the authors of [8] we learned that the results reported in their paper are erroneous and they do not obtain results better than [13].

# References

[1] Francis Bach. Exploring large feature spaces with hierarchical multiple kernel learning. In *NIPS*, 2008.

[2] A. Bosch, A. Zisserman, and X. Munoz. Scene classification via plsa. In *ECCV*, 2006.

[3] Steve Branson, Catherine Wah, Florian Schroff, Boris Babenko, Peter Welinder, Pietro Perona, and Serge Belongie. Visual recognition with humans in the loop. In *ECCV*, 2010.

[4] G. Csurka, C. Bray, C. Dance, and L. Fan. Visual categorization with bags of keypoints. In *Workshop on Statistical Learning in Computer Vision, ECCV*, 2004.

[5] Inderjit Dhillon, Subramanyam Mallela, and Rahul Kumar. A divisive information-theoretic feature clustering algorithm for text classification. *Journal of Machine Learning Research (JMLR)*, 3:1265–1287, 2003.

[6] Noha M. Elfiky, Fahad Shahbaz Khan, Joost van de Weijer, and Jordi Gonzalez. Discriminative compact pyramids for object and scene recognition. *Pattern Recgnition*, 2011.

[7] Brian Fulkerson, Andrea Vedaldi, and Stefano Soatto. Localizing objects with smart dictionaries. In *ECCV*, 2008.

[8] Satoshi Ito and Susumu Kubota. Object classification using hetrogeneous co-occurrence features. In *ECCV*, 2010.

[9] Christopher Kanan and Garrison Cottrell. Robust classification of objects, faces, and flowers using natural image statistics. In *CVPR*, 2010.

[10] Fahad Shahbaz Khan, Joost van de Weijer, and Maria Vanrell. Top-down color attention for object recognition. In *ICCV*, 2009.

[11] Svetlana Lazebnik, Cordelia Schmid, and Jean Ponce. Beyond bags of features: Spatial pyramid matching for recognizing natural scene categories. In *CVPR*, 2006.

[12] D. G. Lowe. Distinctive image features from scale-invariant points. *IJCV*, 60(2):91–110, 2004.

[13] M-E Nilsback and A. Zisserman. Automated flower classification over a large number of classes. In *ICVGIP*, 2008.

[14] Alain Rakotomamonjy, Francis Bach, Stephane Canu, and Yves Grandvalet. More efficiency in multiple kernel learning. In *ICML*, 2007.

[15] J. Sivic, B. Russell, A. Efros, A. Zisserman, and W.Freeman. Discovering object categories in image collections. In *ICCV*, 2005.

[16] Noam Slonim and Naftali Tishby. Agglomerative information bottleneck. In *NIPS*, 1999.

[17] Anne Treisman. Feature Binding, Attention and Object Perception. *Philosophical Transactions: Biological Sciences*, 353(1373):1295–1306, 1998.

[18] Koen E. A. van de Sande, Theo Gevers, and Cees G. M. Snoek. Evaluating color descriptors for object and scene recognition. *PAMI*, 32(9):1582–1596, 2010.

[19] J. van de Weijer, C. Schmid, Jakob J. Verbeek, and D. Larlus. Learning color names for real-world applications. *IEEE Transaction in Image Processing (TIP)*, 18(7):1512–1524, 2009.

[20] Manik Varma and Bodla Rakesh Babu. More generality in efficient multiple kernel learning. In *ICML*, 2009.

[21] Manik Varma and Debajyoti Ray. Learning the discriminative power-invariance trade-off. In *ICCV*, 2007.

[22] Jinjun Wang, Jianchao Yang, Kai Yu, Fengjun Lv, Thomas Huang, and Yihong Gong. Locality-constrained linear coding for image classification. In *CVPR*, 2010.

[23] Bangpeng Yao, Aditya Khosla, and Li Fei-Fei. Combining randomization and discrimination for fine-grained image categorization. In *CVPR*, 2011.

[24] J. Zhang, M. Marszalek, S. Lazebnik, and C. Schmid. Local features and kernels for classification of texture and object catergories: A comprehensive study. *IJCV*, 73(2):213–218, 2007.

